# Four-legged Walking Gait Control Using a Neuromorphic Chip Interfaced to a Support Vector Learning Algorithm

**Susanne Still**
NEC Research Institute
4 Independence Way, Princeton NJ 08540, USA
*sasa@research.nj.nec.com*

**Bernhard Schölkopf**
Microsoft Research Institute
1 Guildhall Street, Cambridge, UK
*bsc@scientist.com*

**Klaus Hepp**
Institute of Theoretical Physics
ETH Zürich, Switzerland

**Rodney J. Douglas**
Institute of Neuroinformatics
ETH/UNI Zürich, Switzerland

## Abstract

To control the walking gaits of a four-legged robot we present a novel neuromorphic VLSI chip that coordinates the relative phasing of the robot's legs similar to how spinal Central Pattern Generators are believed to control vertebrate locomotion [3]. The chip controls the leg movements by driving motors with time varying voltages which are the outputs of a small network of coupled oscillators. The characteristics of the chip's output voltages depend on a set of input parameters. The relationship between input parameters and output voltages can be computed analytically for an idealized system. In practice, however, this ideal relationship is only approximately true due to transistor mismatch and offsets. Fine tuning of the chip's input parameters is done automatically by the robotic system, using an unsupervised Support Vector (SV) learning algorithm introduced recently [7]. The learning requires only that the description of the desired output is given. The machine learns from (unlabeled) examples how to set the parameters to the chip in order to obtain a desired motor behavior.

## 1 Introduction

Modern robots still lag far behind animals in their capability for legged locomotion. Four-legged animals use distinct walking gaits [1], resulting for example in reduction of energy consumption at high speeds [5]. Similarly, the use of different gaits can allow legged robots to adjust their walking behavior not only for speed but also to the terrain they encounter. Coordinating the rhythmic movement patterns necessary for locomotion is a difficult task involving a large number of mechanical degrees of freedom (DOF) and input from many sensors, and considerable advantages may be gained by emulating control architectures found in animals. Neuroscientists have found increasingly strong evidence during the past century to support the hypothesis that centers in the nervous system, called Central Pattern Generators (CPGs), generate rhythmic output responsible for coordinating the large number of muscles needed for locomotion [2]. CPGs are influenced by signals from the brain stem and the cerebellum, brain structures in which locomotive adaptation is believed to take place [11]. This architecture greatly simplifies the control problem for the brain. The brain only needs to set the general level of activity, leaving it to the CPG to coordinate

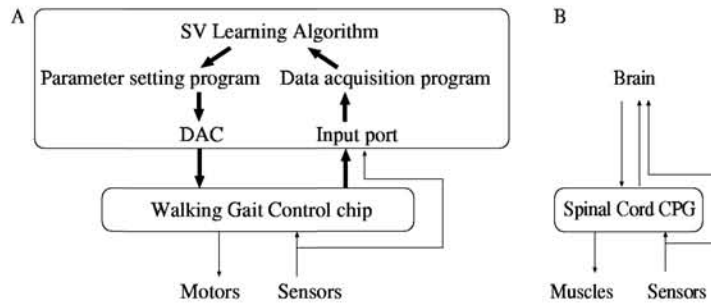

Figure 1: A: Sketch of the control architecture of the robot. Thick arrows indicate the learning loop. B: Sketch of simplified control architecture for locomotion of vertebrates.

the complex pattern of muscle activity required to generate locomotion [3]. We make use of these biological findings by implementing a similar control architecture to control a walking machine. A neuromorphic Gait Controller (GC) chip produces time varying output voltages that control the movement of the robot's legs. Their frequency, phase relationships and duty cycles determine the resulting walking behavior (step frequency, walking gait and direction of motion) and depend on a small set of control parameters. For an ideal system, the relationship between control parameters and output voltages can be determined analytically, but deviations of the chip from ideal have to be compensated for by other means. Since the goal here is that the resulting machine works autonomously, we propose a learning procedure to solve this problem. The robot is given the specifications of the desired movement sequence and explores parameter combinations in the input parameter space of its GC chip, keeping only those leading to a movement that is correct within some tolerance. It then locates the region in input parameter space that contains most of these parameter combinations using an algorithm [7] that extends SV learning to unlabelled data.

## 2 The robotic system

The robotic system consists of (i) a body with one degree of freedom per leg[1] and a po-tentiometer attached to each motor that serves as a sensor providing information about the angular displacement of the leg, (ii) the neuromorphic Gait Controller (GC) chip and (iii) a PC on which algorithms are run to (a) acquire data from chip and sensors, (b) set the chip's input parameters and (c) implement the learning algorithm. The control architecture is inspired by the architecture which is used for locomotion in vertebrates (see Fig. 1 and Sec. 1). Like in biology, the existence of the GC chip considerably simplifies the control task. The computer only needs to set the input parameters to the GC chip, leaving the chip to coordinate the pattern of motor movements necessary to generate locomotion of the robot. The circuitry on the GC chip is based on an analysis [8] of circuits originating from M. W. Tilden (e.g. [4]). The GC chip contains five oscillators which can be inter-connected in different ways. The chip can be used in three reasonable configurations, one of which, a chain of four oscillators (Fig. 2), is used in the present work. Each oscillator (see Fig. 3) consists of two similar sub-circuits. In the following, subscripts $i \in \{1,..,4\}$ will denote the oscillator identity and subscripts $k \in \{l,r\}$ will denote one of the two sub-circuits within an oscillator. Here $l$ stands for the left side of the oscillator circuit and $r$ for the right side. Each sub-circuit has a capacitor connecting an input node to a node $V_{i,k}$, to which the input node of an inverter is connected. The output node of the inverter is called $V_{out,i,k}$.

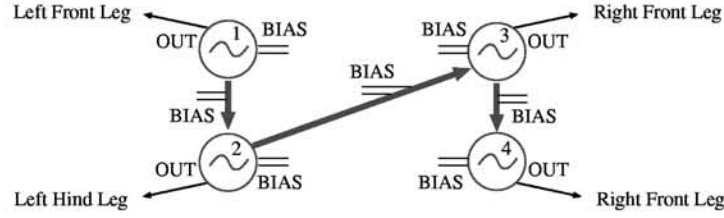

Figure 2: Sketch of the configuration in which the chip is used: a chain of four coupled oscillators. Each of the thin lines is connected to a pad. The round symbols stand for oscillators numbered corresponding to the text. The thick arrows stand for the transmission gates which couple the oscillators (see circuit diagram in Fig. 3). The arrows that lead to the four legs represent the outputs of the oscillators.

Finally, a n-FET transistor with gate voltage $V_{b,i,k}$ is connected between $V_{i,k}$ and ground. An oscillator is obtained by connecting the input node of one sub-circuit to the output node of the other and vice versa. The output voltages of a single oscillator are two mirror-image step functions at $V_{out,i,l}$ and $V_{out,i,r}$. These voltages control the stepping movements of one leg. Two oscillators, $j$ and $j+1$ ($j \in \{1,..,3\}$), are coupled with two transmission gates. One is connected between $V_{out,j,l}$ and $V_{j+1,l}$. The current that flows through it depends on the bias voltage $V_{b,j\ j+1,l}$. Likewise, the other transmission gate connects $V_{out,j,r}$ and $V_{j+1,r}$ and has the bias voltage $V_{b,j\ j+1,r}$. Note that the coupling is asymmetric, affecting only oscillator $j+1$. The voltages at the input nodes to the inverters of oscillator $j$ are not affected by the coupling since the inverters act as impedance buffers.

The chip's output is characterized by the frequency (common to all oscillators), the four duty cycles of the oscillators and three phase lags between oscillators. The phase lags determine which gait the robot adopts. The duty cycles of the oscillators set the ratio between stance and swing phase of the legs. Certain combinations of duty cycles differing from 50% make the robot turn [8]. For a set of constant input parameters, $\{V_{b,i,r}, V_{b,i,l}, V_{b,j\ j+1,r}, V_{b,j\ j+1,l}\}$, a rhythmic output is produced with oscillation period $P$, duty cycles $D_i$ and phase shifts $\phi_j$, where $i \in \{1,..,4\}$ and $j \in \{1,..,3\}$. Analysis of the resulting circuit reveals [8] how the output characteristics of the chip depend on the input parameters. Assume that all transistors on the chip are identical and that the peak voltages at node $V_{1,l}$ and at node $V_{1,r}$ are identical and equal to $V_{max}$. For a certain range of input parameters, the period of the oscillators is given by the period of the first oscillator in the chain (called the master oscillator)

$$P = C(V_{max} - V_{th})(e^{-\frac{q}{kT}\kappa V_{b,1,l}} + e^{-\frac{q}{kT}\kappa V_{b,1,r}})/I_{0n} \tag{1}$$

where $C = 5.159 \times 10^{-10}$ F is the capacitance and $I_{0n}$ is the drain source leakage current of the n-FET. The threshold voltage of the inverter, $V_{th} = 1.345$V, is calculated from the process parameters [9]. $V_{max} = 3.23$V, $I_{0n} = 2.2095 \times 10^{-16}$A and $\kappa = 0.6202$ are estimated with a least squares fit to the data (Fig. 4a). $T$ is the temperature, $k$ the Boltzmann constant and $q$ the electron charge. Let the duty cycle be defined as the fraction of the period during which $V_{out,i,l}$ is high. The master oscillator's duty cycle is

$$D_1 = 1/[1 + e^{\frac{q}{kT}\kappa(V_{b,1,r} - V_{b,1,l})}] \tag{2}$$

A very simple requirement for the controller is to produce a symmetric waveform for straight forward locomotion. For this, all oscillators must have a duty cycle of 1/2 (=50%) [8]. This can be implemented by a symmetric circuit (identical control voltages on both right and left side: $V_{b,j\ j+1,l} = V_{b,j\ j+1,r} =: V_{b,j\ j+1} \ \forall j \in \{1,..,3\}$ and $V_{b,i,l} = V_{b,i,r} =: V_{b,i} \ \forall i \in \{1,..,4\}$). For simplicity, let $V_{b,i} = V_b \ \forall i$. Then the phase lag between oscillators $j$ and $j+1$ is given by (compare Fig. 4b)

$$\phi_j = \frac{1}{2} + \frac{kT/2q(V_{max} - V_{th})}{\beta e^{-\frac{q}{kT}\kappa(V_{b,j\ j+1} + V_b)} - 1} \ln\left[\frac{\gamma(V_{th}) - \mu(V_{th})e^{\frac{q}{kT}\kappa(V_{b,j\ j+1} + V_b)}}{\gamma(V_0) - \mu(V_0)e^{\frac{q}{kT}\kappa(V_{b,j\ j+1} + V_b)}}\right] \tag{3}$$

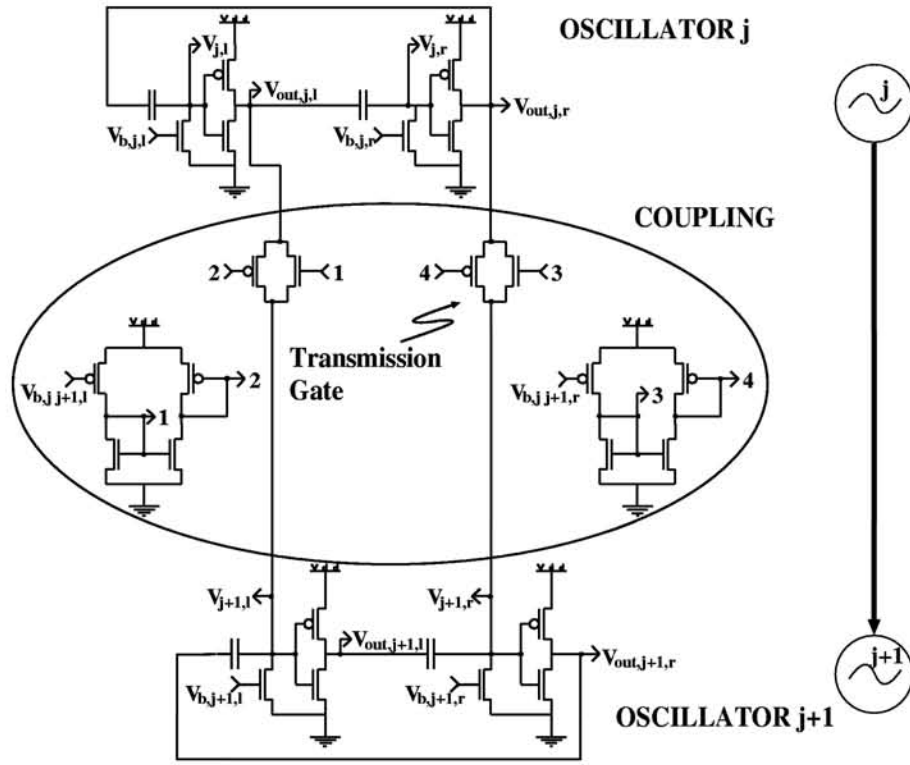

Figure 3: Two oscillators are coupled through a transmission gate. The gate voltage on the n-FET of each transmission gate is set to be the complementary voltage of the p-FET of the same transmission gate by the circuits which are drawn next to the transmission gates. These circuits are controlled by the bias voltages $V_{b,j\,j+1,l}$ and $V_{b,j\,j+1,r}$ and copy the voltages $V_{b,j+1,l}$ and $V_{b,j+1,r}$, to nodes 2 and 4, respectively, while the voltages at nodes 1 and 3 are $(V_{dd} - V_{b,j+1,l})$ and $(V_{dd} - V_{b,j+1,r})$, respectively. The symbols on the right correspond to the symbols in Fig. 2.

where $V_0 = 0.1$ V and

$$\beta = I_{0p}\, e^{\frac{q}{kT}\kappa V_{dd}}/I_{0n}; \quad \gamma(V) = (I_{0n} + I_{0p}\, e^{\frac{q}{kT}V})\, e^{\frac{q}{kT}\kappa V_{dd}}; \quad \mu(V) = I_{0n}\, e^{\frac{q}{kT}V}$$

## 3  Learning

In theory, all duty cycles should be 1/2 for the symmetric circuit. In the real system, the duty cycle changes with the phase lag (see Fig. 4c) due to transistor mismatch. Thus, to obtain a duty cycle of 1/2, $V_{b,j\,j+1,l}$ might have to differ from $V_{b,j\,j+1,r}$. Parameter points which lead to both a desired phase lag and a duty cycle of 1/2 lie in a two dimensional space spanned by $V_{b,j\,j+1,l}$ and $V_{b,j\,j+1,r}$. These parameters are learned[2]. First, a subset $\mathcal{X}$ in this input parameter space is chosen according to the estimate given by (3). $\mathcal{X}$ is scanned and at each point, the output characteristics of the GC chip are determined. If they match the desired output characteristics (within specified tolerances), this point is added to the training data set $\mathcal{V} \subset \mathcal{X}$. After the scan is completed, the training data is transformed by a feature map $\Phi : \mathcal{X} \rightarrow \mathcal{F}$, into a feature space $\mathcal{F}$ such that if $x, y \in \mathcal{X}$, the dot

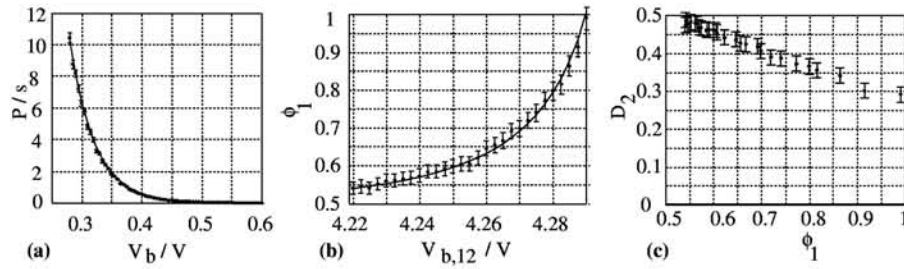

Figure 4: (a): Oscillation period $P$ (points = data, 3% error) as a function of the bias voltage $V_{b,i,l} = V_{b,i,r} =: V_b$. $V_{b,i,l} = V_{b,i,r}$ implies that the duty cycle of the oscillation is 1/2 (see (2)). The oscillation period follows (1) (solid line). (b): Phase lag between the first two oscillators in a chain of 4 oscillators. Function given in (3) (solid line) and data points. $V_{max}$ and $I_{0n}$ are as determined in (a), $I_{0p}$, as estimated by data fitting is $1.56 \times 10^{-19}$ A. (c): Duty cycle of the second oscillator in a chain of 4 oscillators as a function of the phase lag between oscillators 1 and 2.

product $(\Phi(x) \cdot \Phi(y))$ can be computed by evaluation of the Gaussian kernel (which fulfills Mercer's condition [6])

$$k(x,y) = (\Phi(x) \cdot \Phi(y)) = e^{-\|x-y\|^2/2\sigma^2} \qquad (4)$$

In feature space, a hyperplane $(w \cdot \Phi(x)) - \rho = 0$ separating most of the data from the origin with large margin is found by solving the constrained optimization problem (see [7])

$$\min_{w \in \mathcal{F}, \xi \in \mathbb{R}^l, \rho \in \mathbb{R}} \quad \frac{1}{2}\|w\|^2 + \frac{1}{\nu l}\sum_{i=1}^{l} \xi_i - \rho \qquad (5)$$

$$\text{subject to} \quad (w \cdot \Phi(\mathbf{v})) \geq \rho - \xi_i , \quad \xi_i \geq 0 \qquad (6)$$

A decision function, $f$, is computed, which is +1 on a region in input space capturing most of the training data points and -1 elsewhere. The approximate geometrical center of this region is used as the input to the GC chip. The algorithmic implementation of the learning procedure uses the quadratic optimizer LOQO implementing a primal-dual interior-point method [10]. The parameter $\nu$ upper bounds the fraction of outliers (see [7], Proposition 4), which is related to the noise that the training data is subject to. In our experiments, $\nu = 0.2$ is chosen such that the algorithm disregards approximately as many points as can be expected to be falsely included in the training data given the noise of the data acquisition.

## 4  Results

As an example, the input parameters are learned for a forward walk, requiring phase shifts of $\phi_1 = \phi_2 = \phi_3 = 0.75$ and duty cycles of $D_1 = D_2 = D_3 = D_4 = 0.5$. The oscillation period $P = 0.89$s and the duty cycle $D_1 = 0.5$ are set according to (1) and (2). The value of $P$ takes the mechanics of the robot into account [8]. The scanning step size is 2mV and the tolerances are chosen to be $\pm 0.015$ for the phase lags and $\pm 0.05$ for the duty cycles. The parameters $V_{b,j \ j+1,l}$ and $V_{b,j \ j+1,r}$ are learned in sequence, first for $j = 1$ (see Fig. 5). The result is applied to the GC chip. Then $V_{b,23,l}$ and $V_{b,23,r}$ are learned and the result is also applied to the GC chip. Finally, $V_{b,34,l}$ and $V_{b,34,r}$ are learned. All input parameters of the GC chip are set to the learned values and the robot moves forward using a walk gait (see Fig. 6). The phase relationships of the robot's leg movements are measured. Simultaneously, the robot's trajectory is tracked using a video camera monitoring the robot from above, and two Light Emitting Diodes (LEDs) attached to the robot's front and rear. The robot has learned to move in the forward direction, using a walk gait, as desired, despite the inability to theoretically predict the exact values of the GC chip's bias voltages.

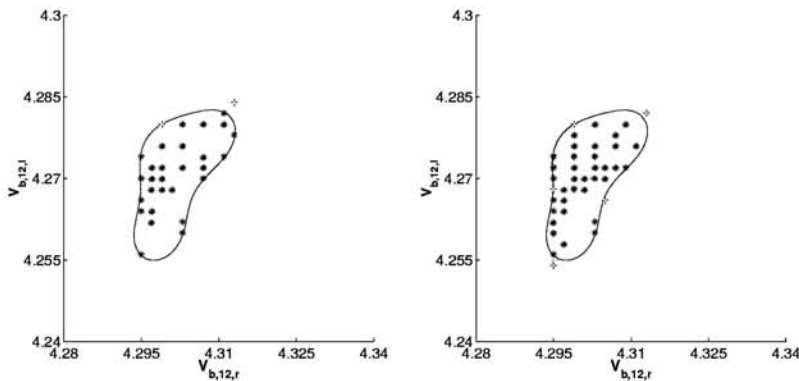

Figure 5: Result of learning values of the bias voltages $V_{b,12,l}$ and $V_{b,12,r}$ which lead to $\phi_1 = 0.75$ and $D_2 = 0.5$. Correctly classified (stars) and misclassified (crosses) training data (left) and test data (right). Outlined regions: learned by algorithm from training data. The training data is obtained from one scan of the displayed rectangular region $\mathcal{X}$. The test data is a set of points obtained from three scans.

## 5   Discussion

We have introduced a novel neuromorphic chip for inter-leg coordination of a walking machine. This chip successfully controls a four-legged robot. A Support Vector algorithm enables the robotic system to learn a desired movement sequence. We have demonstrated this here using the walk gait as an example. Other gaits have also been learned [8]. The architecture we used reduced the learning of a complex motor behavior to a classification task. The classifier we used requires only a few examples, making the learning efficient, and it can handle noisy data, making the learning robust.

The chip need not be interfaced to a computer; it can control the robot without any need of software once the input parameters of the chip are known. Note, that the chips bias voltages can also be changed by simple sensors in a direct way, enabling the robot to adapt its behavior according to sensory information. This point is elaborated in [8].

However, the chip-computer interface creates a hybrid system in which the complex movement pattern required to make a four legged machine locomote is controlled by the chip, while algorithms run on the computer can focus on more demanding tasks. This architecture enables the robotic system to exploit the motor abilities it has due to the GC chip – independent of the particular physical shape of the robot.

The hybrid system could also be useful for the development of a second generation of neuromorphic motor control chips, able to solve more complex tasks. Furthermore, the control circuit could easily be extended to the control of six (or more) legs simply by addition of two (or more) oscillators, without increasing drastically in complexity, as the number of control parameters is small and scales linearly with the number of oscillators. Similarly, the circuit could be expanded to control $n$-jointed legs if each of the four oscillators becomes itself the master of a chain of $n$ oscillators. Finally, the learning procedure introduced here could be used as a general method for fine tuning of neuromorphic aVLSI chips.

**Acknowledgments**

S. S. is grateful to the late Misha Mahowald for inspiring discussions and indebted to Mark W. Tilden for discussing his circuits. We thank Adrian M. Whatley for useful comments and technical assistance. For helpful discussions we thank William Bialek, Gert Cauwenberghs, Giacomo Indiveri, Shih-Chii Liu, John C. Platt, Alex J. Smola, John Shawe-Taylor and Robert C. Williamson. S. S. was supported by CSEM, Neuchâtel, the Physics Department of ETH Zürich and the SPP of the Swiss National Science Foundation.

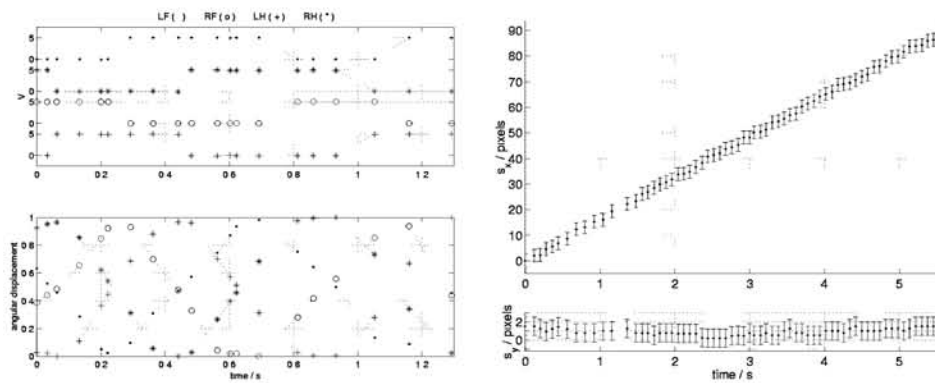

Figure 6: Left: Control voltages (upper plot) and angular displacements of the legs as measured by potentiometers that are attached to the motors (lower plot) as a function of time shown for a cycle of the rhythmic movement. The four legs are distinguished by the abbreviations: left front (LF; dots), right front (RF; circles), left hind (LH; crosses) and right hind (RH; stars). The legs move in succession with a phase shift of $90^o$: LF, RH, RF, and finally LH, a typical walk sequence [1]. Note that the data is acquired with a limited sampling rate. Thus the duty cycles of the control voltages appear to deviate from 50 %. However, the data on the right proves that the duty cycles are sufficiently close to 50 % to cause the robot to walk forward in a straight line, as desired. Right: Position of the robot's center of gravity as a function of time. Upper plot: x-coordinate, lower plot: y-coordinate. Errors are due mainly to the extension of the images of the LEDs in the image frames obtained from the CCD camera. The y-coordinate is constant within the error. This shows that the robot moves forward on a straight line. The robot moves at roughly $4.7 \, \mathrm{cm\,s^{-1}}$.

## Footnotes

[1]The robot's aluminum body is 12 cm long and 6.2 cm wide. It has four DC motors which drive aluminum legs attached at right angles to the plane of the body. Each leg ends in a foot that contains a small electromagnet which is activated during the stance phase of the leg and deactivated during the swing phase. Leg and foot together have a length of 6.5 cm. The robot walks on a metal floor so that the electromagnet increases the friction during the stance phase. For further details see [7].

[2]The desired duty cycle of 1/2 is an example, leading to forward locomotion for of the test robot. In the same way, any other value for the duty cycle can be learned (for examples see [8])

## References

[1] R. McN. Alexander, The Gaits of Bipedal and Quadrupedal Animals. *Intl. J. Robotics Research*, 1984, **3**, pp. 49-59

[2] F. Delcomyn, Neural Basis of Rhythmic Behaviour in Animals. *Science*, 1980, **210**, pp. 492-498

[3] S. Grillner, 1981, Control of locomotion in bipeds, tetrapods and fish. In: *Handbook of Physiology II*, M. D. Bethesda (ed.), Am. Physiol. Soc., pp. 1179-1236; S. Grillner, 1998, Vertebrate Locomotion - A Lamprey Perspective. In: *Neuronal Mechanisms for Generating Locomotor Activity*, O. Kiehn et. al. (eds.), New York Academy of Science

[4] B. Hasslacher & M. W. Tilden, Living Machines. *Robotics and Autonomous Systems: The Biology and Technology of Intelligent Autonomous Agents*, 1995, L. Steels (ed.), Elsevier; S. Still & M. W. Tilden Controller for a four legged walking machine. In: *Neuromorphic Systems*, 1998, L. S. Smith & A. Hamilton (eds.), World Scientific

[5] D. F. Hoyt & R. C. Taylor, Gait and the energetics of locomotion in horses. *Nature*, 1981, **292**, pp. 239-240

[6] J. Mercer, Functions of positive and negative type and their connection with the theory of integral equations. *Phil. Trans. Roy. Soc. London A*, 1909, **209**, pp. 415-446

[7] B. Schölkopf, J. C. Platt, J. Shawe-Taylor, A. J. Smola and R. C. Williamson, Estimating the Support of a High-Dimensional Distribution. Technical Report, Microsoft Research, 1999, MSR-TR-99-87, Redmond, WA, To appear in *Neural Computation*.

[8] S. Still, *Walking Gait Control for Four-legged Robots*, PhD Thesis, ETH Zürich, 2000

[9] N. H. E. Weste & K. Eshraghian, *Principles of CMOS VLSI Design*, 1993, Addison Wesley

[10] R. J. Vanderbei, *LOQO User's Manual – Version 3.10*, Technical Report, SOR-97-08, Princeton University, Statistics and Operations Research, 1997

[11] D. Yanagihara, M. Udo, I. Kondo and T. Yoshida, *Neuroscience Research*, 1993, **18**, pp. 241-244
